# Hot Coupling: A Particle Approach to Inference and Normalization on Pairwise Undirected Graphs of Arbitrary Topology

**Firas Hamze**      **Nando de Freitas**
Department of Computer Science
University of British Columbia

## Abstract

This paper presents a new sampling algorithm for approximating functions of variables representable as undirected graphical models of arbitrary connectivity with pairwise potentials, as well as for estimating the notoriously difficult partition function of the graph. The algorithm fits into the framework of sequential Monte Carlo methods rather than the more widely used MCMC, and relies on constructing a sequence of intermediate distributions which get closer to the desired one. While the idea of using "tempered" proposals is known, we construct a novel sequence of target distributions where, rather than dropping a global temperature parameter, we sequentially couple individual pairs of variables that are, initially, sampled exactly from a spanning tree of the variables. We present experimental results on inference and estimation of the partition function for sparse and densely-connected graphs.

## 1   Introduction

Undirected graphical models are powerful statistical tools having a wide range of applications in diverse fields such as image analysis [1, 2], conditional random fields [3], neural models [4] and epidemiology [5]. Typically, when doing inference, one is interested in obtaining the *local beliefs*, that is the marginal probabilities of the variables given the evidence set. The methods used to approximate these intractable quantities generally fall into the categories of *Markov Chain Monte Carlo* (MCMC) [6] and *variational methods* [7]. The former, involving running a Markov chain whose invariant distribution is the distribution of interest, can suffer from slow convergence to stationarity and high correlation between samples at stationarity, while the latter is not guaranteed to give the right answer or always converge. When performing learning in such models however, a more serious problem arises: the parameter update equations involve the normalization constant of the joint model at the current value of parameters, from here on called the *partition function*. MCMC offers no obvious way of approximating this wildly intractable sum [5, 8]. Although there exists a polynomial time MCMC algorithm for simple graphs with binary nodes, ferromagnetic potentials and *uniform observations* [9], this algorithm is hardly applicable to the complex models encountered in practice. Of more interest, perhaps, are the theoretical results that show that Gibbs sampling and even Swendsen-Wang[10] can mix exponentially slowly in many situations [11]. This paper introduces a new sequential Monte Carlo method for approximating expectations of a *pairwise* graph's variables (of which beliefs are a special case) and of reasonably estimating the partition function. Intuitively, the new method uses interacting parallel chains to handle multimodal distributions,

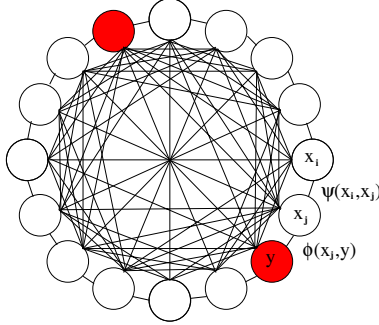

Figure 1: A small example of the type of graphical model treated in this paper. The observations correspond to the two shaded nodes.

with communicating chains distributed across the modes. In addition, there is no requirement that the chains converge to equilibrium as the bias due to incomplete convergence is corrected for by importance sampling.

Formally, given hidden variables $\mathbf{x}$ and observations $\mathbf{y}$, the model is specified on a graph $\mathcal{G}(\mathcal{V}, \mathcal{E})$, with edges $\mathcal{E}$ and $M$ nodes $\mathcal{V}$ by:

$$\pi(\mathbf{x}, \mathbf{y}) = \frac{1}{Z} \prod_{i \in \mathcal{V}} \phi(x_i, y_i) \prod_{(i,j) \in \mathcal{E}} \psi(x_i, x_j)$$

where $\mathbf{x} = \{x_1, \ldots, x_M\}$, $Z$ is the partition function, $\phi(\cdot)$ denotes the observation potentials and $\psi(\cdot)$ denotes the pair-wise interaction potentials, which are strictly positive but otherwise arbitrary. The partition function is: $Z = \sum_{\mathbf{x}} \prod_{i \in \mathcal{V}} \phi(x_i, y_i) \prod_{(i,j) \in \mathcal{E}} \psi(x_i, x_j)$, where the sum is over all possible system states. We make no assumption about the graph's topology or sparseness, an example is in Figure 1. We present experimental results on both *fully-connected* graphs (cases where each node neighbors every other node) and sparse graphs.

Our approach belongs to the framework of *Sequential Monte Carlo* (SMC), which has its roots in the seminal paper of [12]. Particle filters are a well-known instance of SMC methods [13]. They apply naturally to dynamic systems like tracking. Our situation is different. We introduce artificial dynamics simply as a constructive strategy for obtaining samples of a sequence of distributions converging to the distribution of interest. That is, initially we sample from and easy-to-sample distribution. This distribution is then used as a proposal mechanism to obtain samples from a slightly more complex distribution that is closer to the target distribution. The process is repeated until the sequence of distributions of increasing complexity reaches the target distribution. Our algorithm has connections to a general annealing strategy proposed in the physics [14] and statistics [15] literature, known as *Annealed Importance Sampling* (AIS). AIS is a special case of the general SMC framework [16]. The term *annealing* refers to the lowering of a "temperature parameter," the process of which makes the joint distribution more concentrated on its modes, whose number can be massive for difficult problems. The celebrated *simulated annealing* (SA) [17] algorithm is an *optimization* method relying on this phenomenon; presently, however we are interested in *integration* and so SA does not apply here.

Our approach does not use a global temperature, but sequentially introduces *dependencies* among the variables; graphically, this can be understood as "adding edges" to the graph. In this paper, we restrict ourselves to discrete state-spaces although the method applies to arbitrary continuous distributions.

For our initial distribution we choose a spanning tree of the variables, on which analytic marginalization, *exact* sampling, and computation of the partition function are easily done. After drawing a *population* of samples (*particles*) from this distribution, the sequential phase begins: an edge of the desired graph is chosen and gradually added to the current one as shown in Figure 2. The particles then follow a trajectory according to some proposal

mechanism. The "fitness" of the particles is measured via their importance weights. When the set of samples has become skewed, that is with some containing high weights and many containing low ones, the particles are *resampled* according to their weights. The sequential structure is thus imposed by the propose-and-resample mechanism rather than by any property of the original system. The algorithm is formally described after an overview of SMC and recent work presenting a unifying framework of the SMC methodology outside the context of Bayesian dynamic filtering[16].

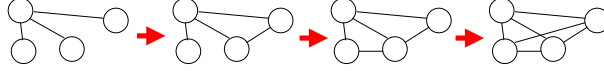

Figure 2: A graphical illustration of our algorithm. First we construct a spanning tree, of which a population of iid samples can be easily drawn using the forward filtering/backward sampling algorithm for trees. The tree then becomes the proposal mechanism for generating samples for a graph with an extra potential. The process is repeated until we obtain samples from the target distribution (defined on a fully connected graph in this case). Edges can be added "slowly" using a coupling parameter.

## 2   Sequential Monte Carlo

As shown in Figure 2, we consider a sequence of auxiliary distributions $\widetilde{\pi}_1(\mathbf{x}_1), \widetilde{\pi}_2(\mathbf{x}_{1:2}), \ldots, \widetilde{\pi}_n(\mathbf{x}_{1:n})$, where $\widetilde{\pi}_1(\mathbf{x}_1)$ is the distribution on the weighted spanning tree. The sequence of distributions can be constructed so that it satisfies $\widetilde{\pi}_n(\mathbf{x}_{1:n}) = \pi_n(\mathbf{x}_n)\widetilde{\pi}_n(\mathbf{x}_{1:n-1}|\mathbf{x}_{1:n})$. Marginalizing over $\mathbf{x}_{1:n-1}$ gives us the target distribution of interest $\pi_n(\mathbf{x}_n)$ (the distribution of the graphical model that we want to sample from as illustrated in Figure 2 for $n = 4$). So we first focus on sampling from the sequence of auxiliary distributions. The joint distribution is only known up to a normalization constant: $\widetilde{\pi}_n(\mathbf{x}_{1:n}) = Z_n^{-1} f_n(\mathbf{x}_{1:n})$, where $Z_n \triangleq \int f_n(\mathbf{x}_{1:n})d\mathbf{x}_{1:n}$ is the partition function. We are often interested in computing this partition function and other expectations, such as $I(g(\mathbf{x}_n)) = \int g(\mathbf{x}_n)\pi_n(\mathbf{x}_n)d\mathbf{x}_n$, where $g$ is a function of interest (e.g. $g(\mathbf{x}) = \mathbf{x}$ if we are interested in computing the mean of $\mathbf{x}$).

If we had a set of samples $\{\mathbf{x}_{1:n}^{(i)}\}_{i=1}^N$ from $\widetilde{\pi}$, we could approximate this integral with the following Monte Carlo estimator: $\widehat{\widetilde{\pi}}_n(d\mathbf{x}_{1:n}) = \frac{1}{N}\sum_{i=1}^N \delta_{\mathbf{x}_{1:n}^{(i)}}(d\mathbf{x}_{1:n})$, where $\delta_{\mathbf{x}_{1:n}^{(i)}}(d\mathbf{x}_{1:n})$ denotes the delta Dirac function, and consequently approximate any expectations of interest. These estimates converge almost surely to the true expectation as $N$ goes to infinity. It is typically hard to sample from $\widetilde{\pi}$ directly. Instead, we sample from a proposal distribution $q$ and weight the samples according to the following importance ratio

$$w_n = \frac{f_n(\mathbf{x}_{1:n})}{q_n(\mathbf{x}_{1:n})} = \frac{f_n(\mathbf{x}_{1:n})}{q_n(\mathbf{x}_{1:n})}\frac{q_{n-1}(\mathbf{x}_{1:n-1})}{f_{n-1}(\mathbf{x}_{1:n-1})}w_{n-1}$$

The proposal is constructed sequentially: $q(\mathbf{x}_{1:n}) = q_{n-1}(\mathbf{x}_{1:n-1})q_n(\mathbf{x}_n|\mathbf{x}_{1:n-1})$. Hence, the importance weights can be updated recursively

$$w_n = \frac{f_n(\mathbf{x}_{1:n})}{q_n(\mathbf{x}_n|\mathbf{x}_{1:n-1})f_{n-1}(\mathbf{x}_{1:n-1})}w_{n-1} \tag{1}$$

Given a set of $N$ particles $\mathbf{x}_{1:n-1}^{(i)}$, we obtain a set of particles $\mathbf{x}_n^{(i)}$ by sampling from $q_n(\mathbf{x}_n|\mathbf{x}_{1:n-1}^{(i)})$ and applying the weights of equation (1). To overcome slow drift in the particle population, a resampling (selection) step chooses the fittest particles (see the introductory chapter in [13] for a more detailed explanation). We use a state-of-the-art minimum variance resampling algorithm [18].

The ratio of successive partition functions can be easily estimated using this algorithm as follows:

$$\frac{Z_n}{Z_{n-1}} = \frac{\int f_n(\mathbf{x}_{1:n})d\mathbf{x}_{1:n}}{Z_{n-1}} = \int \widehat{w}_n \; \widetilde{\pi}_{n-1}(\mathbf{x}_{1:n-1})q_n(\mathbf{x}_n|\mathbf{x}_{1:n-1})d\mathbf{x}_{1:n} \approx \sum_{i=1}^N \widehat{w}_n^{(i)}\widetilde{w}_{n-1}^{(i)},$$

where $\widetilde{w}_{n-1}^{(i)} = w_{n-1}^{(i)}/\sum_j w_{n-1}^{(j)}$, $\widehat{w}_n = \frac{f_n(\mathbf{x}_{1:n})}{q_n(\mathbf{x}_n|\mathbf{x}_{1:n-1})f_{n-1}(\mathbf{x}_{1:n-1})}$ and $Z_1$ can be easily computed as it is the partition function for a tree.

We can choose a (non-homogeneous) Markov chain with transition kernel $K_n(\mathbf{x}_{n-1}, \mathbf{x}_n)$ as the proposal distribution $q_n(\mathbf{x}_n|\mathbf{x}_{1:n-1})$. Hence, given an initial proposal distribution $q_1(\cdot)$, we have *joint* proposal distribution at step $n$: $q_n(\mathbf{x}_{1:n}) = q_1(\mathbf{x}_1)\prod_{k=2}^{n} K_k(\mathbf{x}_{k-1}, \mathbf{x}_k)$. It is convenient to assume that the artificial distribution $\widetilde{\pi}_n(\mathbf{x}_{1:n-1}|\mathbf{x}_n)$ is also the product of (backward) Markov kernels: $\widetilde{\pi}_n(\mathbf{x}_{1:n-1}|\mathbf{x}_n) = \prod_{k=1}^{n-1} L_k(\mathbf{x}_{k+1}, \mathbf{x}_k)$ [16]. Under these choices, the (unnormalized) incremental importance weight becomes:

$$w_n \propto \frac{f_n(\mathbf{x}_n)L_{n-1}(\mathbf{x}_n, \mathbf{x}_{n-1})}{f_{n-1}(\mathbf{x}_{n-1})K_n(\mathbf{x}_{n-1}, \mathbf{x}_n)} \qquad (2)$$

Different choices of the backward Kernel $L$ result in different algorithms [16]. For example, the choice: $L_{n-1}(\mathbf{x}_n, \mathbf{x}_{n-1}) = \frac{f_n(\mathbf{x}_{n-1})K_n(\mathbf{x}_{n-1}, \mathbf{x}_n)}{f_n(\mathbf{x}_n)}$ results in the AIS algorithm, with weights $w_n \propto \frac{f_n(\mathbf{x}_{n-1})}{f_{n-1}(\mathbf{x}_{n-1})}$. However, we should point out that this method is more general as one can carry out resampling. Note that in this case, the importance weights do not depend on $\mathbf{x}_n$ and, hence, it is possible to do resampling before the importance sampling step. This often leads to huge reduction in estimation error [19]. Also, note that if there are big discrepancies between $f_n(\cdot)$ and $f_{n-1}(\cdot)$ the method might perform poorly. To overcome this, [16] use variance results to propose a different choice of backward kernel, which results in the following incremental importance weights:

$$w_n \propto \frac{f_n(\mathbf{x}_n)}{\int f_{n-1}(\mathbf{x}_{n-1})K_n(\mathbf{x}_{n-1}, \mathbf{x}_n)d\mathbf{x}_{n-1}} \qquad (3)$$

The integral in the denominator can be evaluated when dealing with Gaussian or reasonable discrete networks.

## 3   The new algorithm

We could try to perform traditional importance sampling by seeking some proposal distribution for the entire graph. This is very difficult and performance degrades *exponentially* in dimension if the proposal is mismatched [20]. We propose, however, to use the samples from the tree distribution (which we call $\pi_0$) as candidates to an intermediate target distribution, consisting of the tree along with a "weak" version of a potential corresponding to some edge of the original graph. Given a set of edges $G_0$ which form a spanning tree of the target graph, we can can use the belief propagation equations [21] and *bottom-up propagation, top-down sampling* [22], to draw a set of $N$ *independent* samples from the tree. Computation of the normalization constant $Z_1$ is also straightforward and efficient in the case of trees using a sum-product recursion. From then on, however, the normalization constants of subsequent target distributions cannot be analytically computed.

We then choose a new edge $e_1$ from the set of "unused" edges $\mathcal{E} - G_0$ and add it to $G_0$ to form the new edge set $G_1 = e_1 \cup G_0$. Let the vertices of $e_1$ be $u_1$ and $v_1$. Then, the intermediate target distribution $\pi_1$ is proportional to $\pi_0(\mathbf{x}_1)\psi_{e_1}(x_{u_1}, x_{v_1})$. In doing straightforward importance sampling, using $\pi_0$ as a proposal for $\pi_1$, the importance weight is proportional to $\psi_{e_1}(x_{u_1}, x_{v_1})$. We adopt a slow proposal process to move the population of particles towards $\pi_1$. We gradually introduce the potential between $X_{u_1}$ and $X_{v_1}$ via a *coupling parameter* $\alpha$ which increases from 0 to 1 in order to "softly" bring the edge's potential in and allow the particles to adjust to the new environment. Formally, when adding edge $e_1$ to the graph, we introduce a number of coupling steps so that we have the intermediate target distribution:

$$\pi_0(\mathbf{x}_0)\left[\psi_{e_1}(x_{u_1}, x_{v_1})\right]^{\alpha_n}$$

where $\alpha_n$ is defined to be 0 when a new edge enters the sequence, increases to 1 as the edge is brought in, and drops back to zero when another edge is added at the following edge iteration.

At each time step, we want a proposal mechanism that is close to the target distribution. Proposals based on simple perturbations, such as random walks, are easy to implement, but can be inefficient. Metropolis-Hastings proposals are not possible because of the integral in the rejection term. We can, however, employ a single-site Gibbs sampler with random scan whose invariant distribution at each step is the *the next target density* in the sequence; this kernel is applied to each particle. When an edge has been fully added a new one is chosen and the process is repeated until the final target density is the full graph. We use an analytic expression for the incremental weights corresponding to Equation (3).

To alleviate potential confusion with MCMC, while any one particle obviously forms a correlated path, we are using a population and are making no assumption or requirement that the chains have converged as is done in MCMC as we are correcting for incomplete convergence with the weights.

## 4   Experiments and discussion

Four approximate inference methods were compared: our SMC method with sequential edge addition (Hot Coupling (HC)), a more typical annealing strategy with a global temperature parameter(SMCG), single-site Gibbs sampling with random scan and loopy belief propagation. SMCG can be thought of as related to HC but where all the edges and local evidence are annealed at the same time.

The majority of our experiments were performed on graphs that were small enough for exact marginals and partition functions to be exhaustively calculated. However, even in toy cases MCMC and loopy can give unsatisfactory and sometimes disastrous results. We also ran a set of experiments on a relatively large MRF.

For the small examples we examined both fully-connected (FC) and square grid (MRF) networks, with 18 and 16 nodes respectively. Each variable could assume one of 3 states. Our pairwise potentials corresponded to the well-known *Potts model*: $\psi_{i,j}(x_i, x_j) = e^{\frac{1}{T} J_{ij} \delta_{x_i, x_j}}$, $\phi_i(x_i) = e^{\frac{1}{T} J \delta_{x_i}(y_i)}$. We set $T = 0.5$ (a low temperature) and tested models with uniform and positive $J_{ij}$, widely used in image analysis, and models with $J_{ij}$ drawn from a standard Gaussian; the latter is an instance of the much-studied *spin-glass* models of statistical physics which are known to be notoriously difficult to simulate at low temperatures [23]. Of course fully-connected models are known as *Boltzmann machines* [4] to the neural computation community. The output potentials were randomly selected in both the uniform and random interaction cases. The HC method used a linear coupling schedule for each edge, increasing from $\alpha = 0$ to $\alpha = 1$ over 100 iterations; our SMCG implementation used a linear global cooling schedule, whose number of steps depended on the graph in order to match those taken by SMCG.

All Monte Carlo algorithms were independently run 50 times each to approximate the variance of the estimates. Our SMC simulations used 1000 particles for each run, while each Gibbs run performed 20000 single-site updates. For these models, this was more than enough steps to settle into local minima; runs of up to 1 million iterations did not yield a difference, which is characteristic of the exponential mixing time of the sampler on these graphs. For our HC method, *spanning trees and edges in the sequential construction were randomly chosen from the full graph;* the rationale for doing so is to allay any criticism that "tweaking" the ordering may have had a crucial effect on the algorithm. The order clearly would matter to some extent, but this will be examined in later work. Also in the tables by "error" we mean the quantity $\frac{|\hat{a}-a|}{a}$ where $\hat{a}$ is an estimate of some quantity $a$ obtained exactly (say $Z$).

First, we used HC, SMCG and Gibbs to approximate the expected sum of our graphs' variables, the so-called *magnetization*: $m = E[\sum_{i=1}^{M} x_i]$. We then approximated the partition functions of the graphs using HC, SMCG, and loopy.[1] We note again that there is no obvious way of estimating $Z$ using Gibbs. Finally, we approximated the marginal probabilities using the four approximate methods. For loopy, we only kept the runs where it converged.

| Method | MRF Random $\Psi$ Error | Var | MRF Homogeneous $\Psi$ Error | Var | FC Random $\Psi$ Error | Var | FC Homogeneous $\Psi$ Error | Var |
|---|---|---|---|---|---|---|---|---|
| HC | 0.0022 | 0.012 | 0.0251 | 0.17 | 0.0016 | 0.0522 | 0.0036 | 0.038 |
| SMCG | 0.0001 | 0.03 | 0.2789 | 10.09 | 0.127 | 0.570 | 0.331 | 165.61 |
| Gibbs | 0.0003 | 0.014 | 0.4928 | 200.95 | 0.02 | 0.32 | 0.3152 | 201.08 |

Figure 3: Approximate magnetization for the nodes of the graphs, as defined in the text, calculated using HC, SMCG, and Gibbs sampling and compared to the true value obtained by brute force. Observe the massive variance of Gibbs sampling in some cases.

| Method | MRF Random $\Psi$ Error | Var | MRF Homogeneous $\Psi$ Error | Var | FC Random $\Psi$ Error | Var | FC Homogeneous $\Psi$ Err | Var |
|---|---|---|---|---|---|---|---|---|
| HC | 0.0105 | 0.002 | 0.0227 | 0.001 | 0.0043 | 0.0537 | 0.0394 | 0.001 |
| SMCG | 0.004 | 0.005 | 6.47 | 7.646 | 1800 | 1.24 | 1 | 29.99 |
| loopy | 0.005 | - | 0.155 | - | 1 | - | 0.075 | - |

Figure 4: Approximate partition function of the graphs discussed in the text calculated using HC, SMCG, and Loopy Belief Propagation (loopy.) For HC and SMCG are shown the error of the sample average of results over 50 independent runs and the variance across those runs. loopy is of course a deterministic algorithm and has no variance. HC maintains a low error and variance in all cases.

Figure 3 shows the results of the magnetization experiments. On the MRF with random interactions, all three methods gave very accurate answers with small variance, but for the other graphs, the accuracies and variances began to diverge. On both positive-potential graphs, Gibbs sampling gives high error and huge variance; SMCG gives lower variance but is still quite skewed. On the fully-connected random-potential graph the 3 methods give good results but HC has the lowest variance. Our method experiences its worst performance on the homogeneous MRF but *it is only 2.5% error!*

Figure 4 tabulates the approximate partition function calculations. Again, for the MRF with random interactions, the 3 methods give estimates of $Z$ of comparable quality. This example appeared to work for loopy, Gibbs, and SMCG. For the homogeneous MRF, SMCG degrades rapidly; loopy is still satisfactory at 15% error, but HC is at 2.7% with very low variance. In the fully-connected case with random potentials, HC's error is 0.43% while loopy's error is very high, having underestimated $Z$ by a factor of $10^5$. SMCG fails completely here as well. On the uniform fully-connected graph, loopy actually gives a reasonable estimate of $Z$ at 7.5%, but is still beaten by HC.

Figure 5 shows the *variational* ($L_1$) distance between the exact marginal for a randomly chosen node in each graph and the approximate marginals of the 4 algorithms, a common measure of the "distance" between 2 distributions. For the Monte Carlo methods (HC, SMCG and Gibbs) the average over 50 independent runs was used to approximate the *expected* $L_1$ error of the estimate. All 4 methods perform well on the random $\Psi$ MRF. On the MRF with homogeneous $\Psi$, both loopy and SMCG degrade, but HC maintains a low error. Among the FC graphs, HC performs extremely well on the homogeneous $\Psi$ and surprisingly loopy does well too. In the random $\Psi$ case, loopy's error increases dramatically.

Our final set of simulations was the classic Mean Squared reconstruction of a noisy image problem; we used a 100x100 MRF with a noisy "patch" image (consisting of shaded, rectangular regions) with an isotropic 5-state prior model. The object was to calculate the pixels' posterior marginal expectations. We chose this problem because it is a large model on which loopy is known to do well on, and can hence provide us with a measure of quality of the HC and SMCG results as larger numbers of edges are involved. From the toy examples we infer that the mechanism of HC is quite different from that of loopy as we have seen that it can work when loopy does not. Hence good performance on this problem would suggest that HC would *scale well,* which is a crucial question as in the large graph the final distribution has many more edges than the initial spanning tree. The results were promising: the mean-squared reconstruction error using loopy and using HC were virtually identical at $9.067 \times 10^{-5}$ and $9.036 \times 10^{-5}$ respectively, showing that HC seemed to be

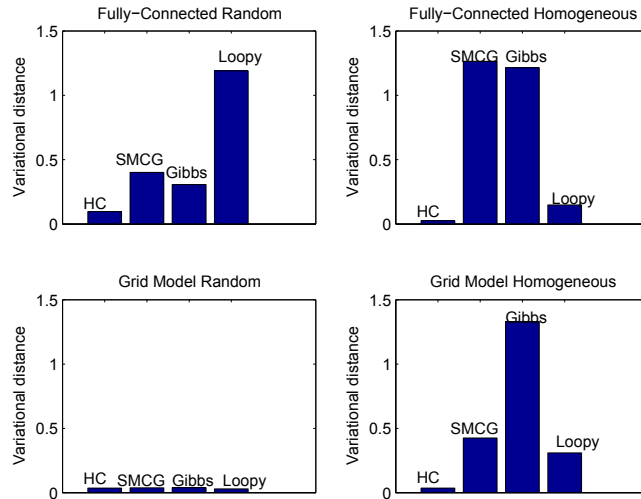

Figure 5: Variational($L_1$) distance between estimated and true marginals for a randomly chosen node in each of the 4 graphs using the four approximate methods (smaller values mean less error.) The MRF-random example was again "easy" for all the methods, but the rest raise problems for all but HC.

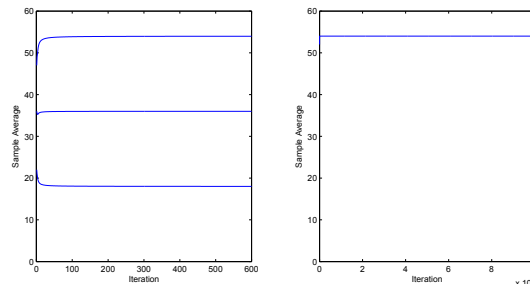

Figure 6: An example of how MCMC can get "stuck:" 3 different runs of a Gibbs sampler estimating the magnetization of FC-Homogeneous graph. At left are shown the first 600 iterations of the runs; after a brief transient behaviour the samplers settled into different minima which persisted for the entire duration (20000 steps) of the runs. Indeed for 1 million steps the local minima persist, as shown at right.

robust to the addition of around 9000 edges and many resampling stages. SMCG on the large MRF did not fare as well.

It is crucial to realize that MCMC is completely unsuited to some problems; see for example the "convergence" plots of the estimated magnetization of 3 independent Gibbs sampler runs on one of our "toy" graphs shown in Figure 6. Such behavior has been studied by Gore and Jerrum [11] and others, who discuss pessimistic theoretical results on the mixing properties of both Gibbs sampling and the celebrated Swendsen-Wang algorithm in several cases. To obtain a good estimate, MCMC requires that the process "visit" each of the target distribution's basins of energy with a frequency representative of their probability. Unfortunately, some basins take an exponential amount of time to exit, and so different finite runs of MCMC will give quite different answers, leading to tremendous variance. The methodology presented here is an attempt to sidestep the whole issue of mixing by permitting the independent particles to be stuck in modes, but then considering them *jointly* when estimating. In other words, instead of using a *time average,* we estimate using a weighted

*ensemble average.* The object of the sequential phase is to address the difficult problem of constructing a suitable proposal for high-dimensional problems; to this the resampling-based methodology of particle filters was thought to be particularly suited. For the graphs we have considered, the single-edge algorithm we propose seems to be preferable to global annealing.

## Footnotes

[1] Code for Bethe $Z$ approximation kindly provided by Kevin Murphy.

## References

[1] S Z Li. *Markov random field modeling in image analysis.* Springer-Verlag, 2001.

[2] P Carbonetto and N de Freitas. Why can't José read? the problem of learning semantic associations in a robot environment. In *Human Language Technology Conference Workshop on Learning Word Meaning from Non-Linguistic Data*, 2003.

[3] J D Lafferty, A McCallum, and F C N Pereira. Conditional random fields: Probabilistic models for segmenting and labeling sequence data. In *International Conference on Machine Learning*, 2001.

[4] D E Rumelhart, G E Hinton, and R J Williams. Learning internal representations by error propagation. In D E Rumelhart and J L McClelland, editors, *Parallel Distributed Processing: Explorations in the Microstructure of Cognition*, pages 318–362, Cambridge, MA, 1986.

[5] P J Green and S Richardson. Hidden Markov models and disease mapping. *Journal of the American Statistical Association*, 97(460):1055–1070, 2002.

[6] C P Robert and G Casella. *Monte Carlo Statistical Methods.* Springer-Verlag, New York, 1999.

[7] M. I. Jordan, Z. Ghahramani, T. S. Jaakkola, and L. K. Saul. An introduction to variational methods for graphical models. *Machine Learning*, 37:183–233, 1999.

[8] J Moller, A N Pettitt, K K Berthelsen, and R W Reeves. An efficient Markov chain Monte Carlo method for distributions with intractable normalising constants. Technical report, The Danish National Research Foundation: Network in Mathematical Physics and Stochastics, 2004.

[9] M Jerrum and A Sinclair. The Markov chain Monte Carlo method: an approach to approximate counting and integration. In D S Hochbaum, editor, *Approximation Algorithms for NP-hard Problems*, pages 482–519. PWS Publishing, 1996.

[10] R H Swendsen and J S Wang. Nonuniversal critical dynamics in Monte Carlo simulations. *Physical Review Letters*, 58(2):86–88, 1987.

[11] V Gore and M Jerrum. The swendsen-wang process does not always mix rapidly. In *29th Annual ACM Symposium on Theory of Computing*, 1996.

[12] N Metropolis and S Ulam. The Monte Carlo method. *Journal of the American Statistical Association*, 44(247):335–341, 1949.

[13] A Doucet, N de Freitas, and N J Gordon, editors. *Sequential Monte Carlo Methods in Practice.* Springer-Verlag, 2001.

[14] C Jarzynski. Nonequilibrium equality for free energy differences. *Phys. Rev. Lett.*, 78, 1997.

[15] R M Neal. Annealed importance sampling. Technical Report No 9805, University of Toronto, 1998.

[16] P Del Moral, A Doucet, and G W Peters. Sequential Monte Carlo samplers. Technical Report CUED/F-INFENG/2004, Cambridge University Engineering Department, 2004.

[17] S Kirkpatrick, C D Gelatt, and M P Vecchi. Optimization by simulated annealing. *Science*, 220:671–680, 1983.

[18] G Kitagawa. Monte Carlo filter and smoother for non-Gaussian nonlinear state space models. *Journal of Computational and Graphical Statistics*, 5:1–25, 1996.

[19] N de Freitas, R Dearden, F Hutter, R Morales-Menendez, J Mutch, and D Poole. Diagnosis by a waiter and a mars explorer. *IEEE Proceedings*, 92, 2004.

[20] J A Bucklew. *Large Deviation Techniques in Decision, Simulation, and Estimation.* John Wiley & Sons, 1986.

[21] J Pearl. *Probabilistic reasoning in intelligent systems: networks of plausible inference.* Morgan-Kaufmann, 1988.

[22] C K Carter and R Kohn. On Gibbs sampling for state space models. *Biometrika*, 81(3):541–553, 1994.

[23] M E J Newman and G T Barkema. *Monte Carlo Methods in Statistical Physics.* Oxford University Press, 1999.
